# A 'Neural' Network that Learns to Play Backgammon

*G. Tesauro*

Center for Complex Systems Research, University of Illinois
at Urbana-Champaign, 508 S. Sixth St., Champaign, IL 61820

*T. J. Sejnowski*

Biophysics Dept., Johns Hopkins University, Baltimore, MD 21218

## ABSTRACT

We describe a class of connectionist networks that have learned to play back-gammon at an intermediate-to-advanced level. The networks were trained by a supervised learning procedure on a large set of sample positions evaluated by a human expert. In actual match play against humans and conventional computer programs, the networks demonstrate substantial ability to generalize on the basis of expert knowledge. Our study touches on some of the most important issues in network learning theory, including the development of efficient coding schemes and training procedures, scaling, generalization, the use of real-valued inputs and outputs, and techniques for escaping from local minima. Practical applications in games and other domains are also discussed.

## INTRODUCTION

A potentially quite useful testing ground for studying issues of knowledge representation and learning in networks can be found in the domain of game playing. Board games such as chess, go, backgammon, and Othello entail considerable sophistication and complexity at the advanced level, and mastery of expert concepts and strategies often takes years of intense study and practice for humans. However, the complexities in board games are embedded in relatively "clean" structured tasks with well-defined rules of play, and well-defined criteria for success and failure. This makes them amenable to automated play, and in fact most of these games have been extensively studied with conventional computer science techniques. Thus, direct comparisons of the results of network learning can be made with more conventional approaches.

In this paper, we describe an application of network learning to the game of backgammon. Backgammon is a difficult board game which appears to be well-suited to neural networks, because the way in which moves are selected is primarily on the basis of pattern-recognition or "judgemental" reasoning, as opposed to explicit "look-ahead," or tree-search computations. This is due to the probabilistic dice rolls in backgammon, which greatly expand the branching factor at each ply in the search (to over 400 in typical positions).

Our learning procedure is a supervised one[1] that requires a database of positions and moves that have been evaluated by an expert "teacher." In contrast, in an unsupervised procedure[2-4] learning would be based on the consequences of a given move (e.g., whether it led to a won or lost position), and explicit teacher instructions would not be required. However, unsupervised learning procedures thus far have been much less efficient at reaching high levels of performance than supervised learning procedures. In part, this advantage of supervised learning can be traced to the higher

quantity and quality of information available from the teacher.

Studying a problem of the scale and complexity of backgammon leads one to confront important general issues in network learning. Amongst the most important are scaling and generalization. Most of the problems that have been examined with connectionist learning algorithms are relatively small scale and it is not known how well they will perform on much larger problems. Generalization is a key issue in learning to play backgammon since it is estimated that there are $10^{20}$ possible board positions, which is far in excess of the number of examples that can be provided during training. In this respect our study is the most severe test of generalization in any connectionist network to date.

We have also identified in this study a novel set of special techniques for training the network which were necessary to achieve good performance. A training set based on naturally occurring or random examples was not sufficient to bring the network to an advanced level of performance. Intelligent data-base design was necessary. Performance also improved when noise was added to the training procedure under some circumstances. Perhaps the most important factor in the success of the network was the method of encoding the input information. The best performance was achieved when the raw input information was encoded in a conceptually significant way, and a certain number of pre-computed features were added to the raw information. These lessons may also be useful when connectionist learning algorithms are applied to other difficult large-scale problems.

## NETWORK AND DATA BASE SET-UP

Our network is trained to *select* moves (i.e. to produce a real-valued score for any given move), rather than to *generate* them. This avoids the difficulties of having to teach the network the concept of move legality. Instead, we envision our network operating in tandem with a pre-processor which would take the board position and roll as input, and produce all legal moves as output. The network would be trained to score each move, and the system would choose the move with the highest network score. Furthermore, the network is trained to produce relative scores for each move, rather than an absolute evaluation of each final position. This approach would have greater sensitivity in distinguishing between close alternatives, and corresponds more closely to the way humans actually evaluate moves.

The current data base contains a total of 3202 board positions, taken from various sources[5]. For each position there is a dice roll and a set of legal moves of that roll from that position. The moves receive commentary from a human expert in the form of a relative score in the range [-100,+100], with +100 representing the best possible move and -100 representing the worst possible move. One of us (G.T.) is a strong backgammon player, and played the role of human expert in entering these scores. Most of the moves in the data base were not scored, because it is not feasible for a human expert to comment on all possible moves. (The handling of these unscored lines of data in the training procedure will be discussed in the following section.)

An important result of our study is that in order to achieve the best performance, the data base of examples must be intelligently designed, rather than haphazardly accumulated. If one simply accumulates positions which occur in actual game play, for example, one will find that certain principles of play will appear over and over again in these positions, while other important principles may be used only rarely. This causes problems for the network, as it tends to "overlearn" the commonly used principles, and not learn at all the rarely used principles. Hence it is necessary to have both an intelligent selection mechanism to reduce the number of over-represented situations, and an intelligent design mechanism to enhance the number of examples which illustrate under-represented situations. This process is described in more detail elsewhere[5].

We use a deterministic, feed-forward network with an input layer, an output layer, and either one or two layers of hidden units, with full connectivity between adjacent layers. (We have tried a number of experiments with restricted receptive fields, and generally have not found them to be useful.) Since the desired output of the network is a single real value, only one output unit is required.

The coding of the input patterns is probably the most difficult and most important design issue. In its current configuration the input layer contains 459 input units. A location-based representation scheme is used, in which a certain number of input units are assigned to each of the 26 locations (24 basic plus White and Black bar) on the board. The input is inverted if necessary so that the network always sees a problem in which White is to play.

An example of the coding scheme used until very recently is shown in Fig. 1. This is essentially a unary encoding of the number of men at each board location, with a few exceptions as indicated in the diagram. This representation scheme worked fairly well, but had one peculiar problem in that after training, the network tended to prefer piling large numbers of men on certain points, in particular White's 5 point (the 20 point in the 1-24 numbering scheme). Fig. 2 illustrates an example of this peculiar behavior. In this position White is to play 5-1. Most humans would play 4-5,4-9 in this position; however, the network chose the move 4-9,19-20. This is actually a bad move, because it reduces White's chances of making further points in his inner board. The fault lies not with the data base used to train the network, but rather with the representation scheme used. In Fig. 1a, notice that unit 12 is turned on whenever the final position is a point, and the number of men is different from the initial position. For the 20 point in particular, this unit will develop strong excitatory weights due to cases in which the initial position is not a point (i.e., the move makes the point). The 20 point is such a valuable point to make that the excitation produced by turning unit 12 on might overwhelm the inhibition produced by the poor distribution of builders.

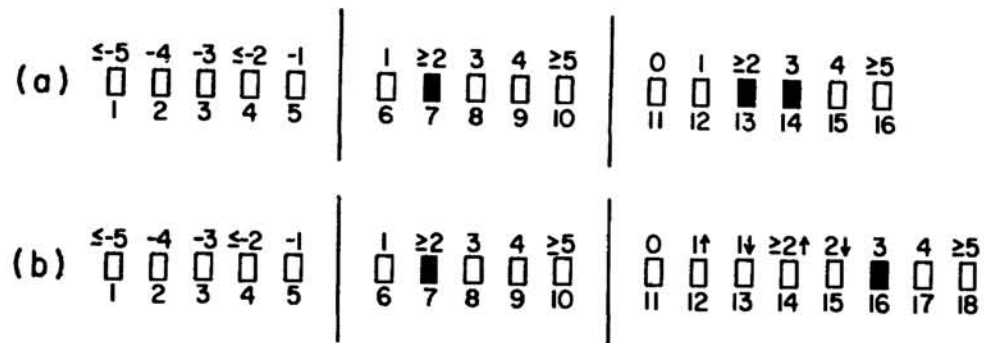

Figure 1-- Two schemes used to encode the raw position information in the network's input. Illustrated in each case is the encoding of two White men present before the move, and three White men present after the move. (a) An essentially unary coding of the number of men at a particular board location. Units 1-10 encode the initial position, units 11-16 encode the final position if there has been a change from the initial position. Units are turned on in the cases indicated on top of each unit, e.g., unit 1 is turned on if there are 5 or more Black men present, etc.. (b) A superior coding scheme with more units used to characterize the type of transition from initial to final position. An up arrow indicates an increase in the number of men, a down arrow indicates a decrease. Units 11-15 have conceptual interpretations: 11=''clearing.'' 12=''slotting,'' 13=''breaking,'' 14=''making,'' 15=''stripping'' a point.

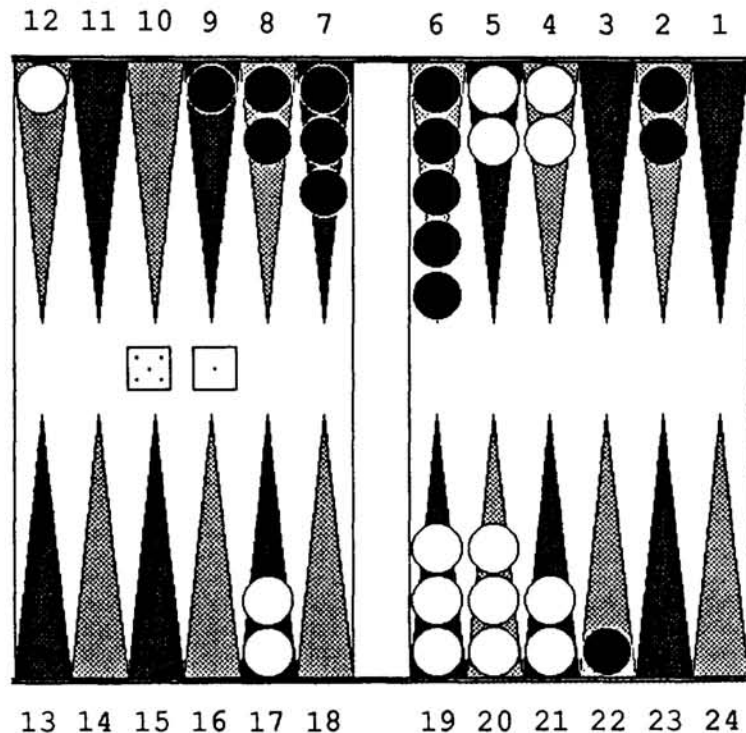

Figure 2-- A sample position illustrating a defect of the coding scheme in Fig. 1a. White is to play 5-1. With coding scheme (1a), the network prefers 4-9, 19-20. With coding scheme (1b), the network prefers 4-9, 4-5. The graphic display was generated on a Sun Microsystems workstation using the Gammontool program.

In conceptual terms, humans would say that unit 12 participates in the representation of two different concepts: the concept of *making* a point, and the concept of *changing* the number of men occupying a made point. These two concepts are unrelated, and there is no point in representing them with a common input unit. A superior representation scheme in which these concepts are separated is shown in Fig. 1b: In this representation unit 13 is turned on only for moves which make the point. Other moves which change the number of men on an already-made point do not activate unit 13, and thus do not receive any undeserved excitation. With this representation scheme the network no longer tends to pile large numbers of men on certain points, and its overall performance is significantly better.

In addition to this representation of the raw board position, we also utilize a number of input units to represent certain "pre-computed" features of the raw input. The principal goal of this study has been to investigate network learning, rather than simply to obtain high performance, and thus we have resisted the temptation of including sophisticated hand-crafted features in the input encoding. However, we have found that a few simple features are needed in practice to obtain minimal standards of competent play. With only "raw" board information, the order of the desired computation (as defined by Minsky and Papert[6]) is probably quite high, and the number of examples needed to learn such a difficult computation might be intractably large. By giving the network "hints" in the form of pre-computed features, this reduces the order of the computation, and thus might make more of the problem learnable in a tractable number of examples.

## TRAINING AND TESTING PROCEDURES

To train the network, we have used the standard "back-propagation" learning algorithm[7-9] for modifying the connections in a multilayer feed-forward network. (A detailed discussion of learning parameters, etc., is provided elsewhere[5].) However, our procedure differs from the standard procedure due to the necessity of dealing with the large number of uncommented moves in the data base. One solution would be simply to avoid presenting these moves to the network. However, this would limit the variety of input patterns presented to the network in training, and certain types of inputs probably would be eliminated completely. The alternative procedure which we have adopted is to skip the uncommented moves most of the time (75% for ordinary rolls and 92% for double rolls), and the remainder of the time present the pattern to the network and generate a random teacher signal with a slight negative bias. This makes sense, because if a move has not received comment by the human expert, it is more likely to be a bad move than a good move. The random teacher signal is chosen uniformly from the interval [-65,+35].

We have used the following four measures to assess the network's performance after it has been trained: (i) performance on the training data, (ii) performance on a set of test data (1000 positions) which was not used to train the network, (iii) performance in actual game play against a conventional computer program (the program *Gammontool* of Sun Microsystems Inc.), and (iv) performance in game play against a human expert (G.T.). In the first two measures, we define the performance as the fraction of positions in which the network picks the correct move, i.e., those positions for which the move scored highest by the network agrees with the choice of the human expert. In the latter two measures, the performance is defined simply as the fraction of games won, without considering the complications of counting gammons or backgammons.

## QUANTITATIVE RESULTS

A summary of our numerical results as measured by performance on the training set and against Gammontool is presented in Table 1. The best network that we have produced so far appears to defeat Gammontool nearly 60% of the time. Using this as a benchmark, we find that the most serious decrease in performance occurs by removing all pre-computed features from the input coding. This produces a network which wins at most about 41% of the time. The next most important effect is the removal of noise from the training procedure; this results in a network which wins 45% of the time. Next in importance is the presence of hidden units; a network without hidden units wins about 50% of the games against Gammontool. In contrast, effects such as varying the exact number of hidden units, the number of layers, or the size of the training set, results in only a few (1-3) percentage point decrease in the number of games won.

Also included in Table 1 is the result of an interesting experiment in which we removed our usual set of pre-computed features and substituted instead the individual terms of the Gammontool evaluation function. We found that the resulting network, after being trained on our expert training set, was able to defeat the Gammontool program by a small margin of 54 to 46 percent. The purpose of this experiment was to provide evidence of the usefulness of network learning as an adjunct to standard AI techniques for hand-crafting evaluation functions. Given a set of features to be used in an evaluation function which have been designed, for example, by interviewing a human expert, the problem remains as to how to "tune" these features, i.e., the relative weightings to associate to each feature, and at an advanced level, the context in which each feature is relevant. Little is known in general about how to approach this problem, and often the human programmer must resort to painstaking trial-and-error tuning by hand. We claim that network learning is a powerful, general-purpose, automated method of approaching this problem, and has the potential to produce a tuning which is superior to those produced by humans, given a data base of sufficiently high quality, and a suitable scheme for encoding the features. The result of our experiment provides evidence to support this claim, although it is not firmly established since we do not have highly accurate statistics, and we do not know how much human effort went into the tuning of the Gammontool evaluation

function. More conclusive evidence would be provided if the experiment were repeated with a more sophisticated program such as Berliner's BKG[10], and similar results were obtained.

| | Network size | Training cycles | Perf. on test set | Perf. vs. Gammontool | Comments |
|---|---|---|---|---|---|
| (a) | 459-24-24-1 | 20 | .540 | .59 ± .03 | |
| (b) | 459-24-1 | 22 | .542 | .57 ± .05 | |
| (c) | 459-24-1 | 24 | .518 | .58 ± .05 | 1600 posn. D.B. |
| (d) | 459-12-1 | 10 | .538 | .54 ± .05 | |
| (e) | 410-24-12-1 | 16 | .493 | .54 ± .03 | Gammontool features |
| (f) | 459-1 | 22 | .485 | .50 ± .03 | No hidden units |
| (g) | 459-24-12-1 | 10 | .499 | .45 ± .03 | No training noise |
| (h) | 393-24-12-1 | 12 | .488 | .41 ± .02 | No features |

Table 1-- Summary of performance statistics for various networks. (a) The best network we have produced, containing two layers of hidden units, with 24 units in each layer. (b) A network with only one layer of 24 hidden units. (c) A network with 24 hidden units in a single layer, trained on a training set half the normal size. (d) A network with half the number of hidden units as in (b). (e) A network with features from the Gammontool evaluation function substituted for the normal features. (f) A network without hidden units. (g) A network trained with no noise in the training procedure. (h) A network with only a raw board description as input.

## QUALITATIVE RESULTS

Analysis of the weights produced by training a network is an exceedingly difficult problem, which we have only been able to approach qualitatively. In Fig. 3 we present a diagram showing the connection strengths in a network with 651 input units and no hidden units. The figure shows the weights from each input unit to the output unit. (For purposes of illustration, we have shown a coding scheme with more units than normal to explicitly represent the transition from initial to final position.) Since the weights go directly to the output, the corresponding input units can be clearly interpreted as having either an overall excitatory or inhibitory effect on the score produced by the network.

A great deal of columnar structure is apparent in Fig. 3. This indicates that the network has learned that a particular number of men at a given location, or a particular type of transition at a given location, is either good or bad independent of the exact location on the board where it occurs. Furthermore, we can see the importance of each of the pre-computed features in the input coding. The most significant features seem to be the number of points made in the network's inner board, and the total blot exposure.

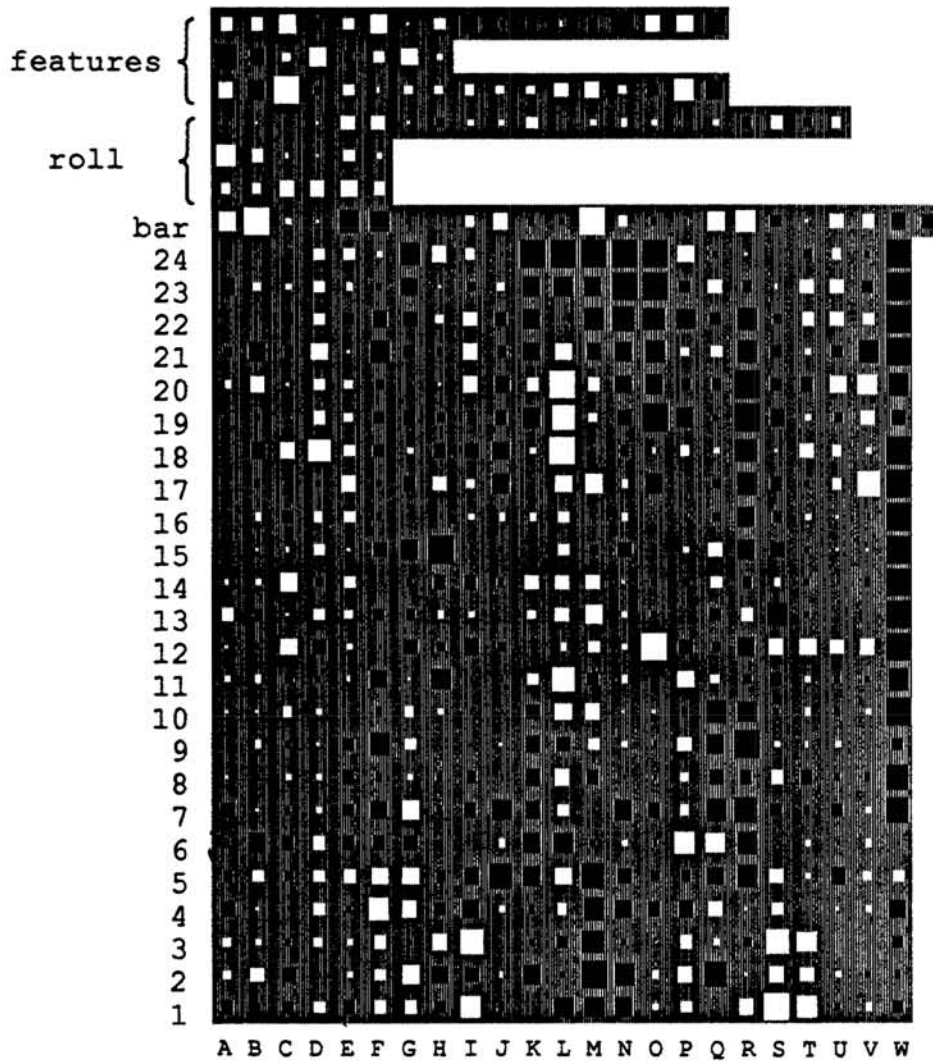

Figure 3-- A Hinton diagram for a network with 651 input units and no hidden units. Small squares indicate weights from a particular input unit to the output unit. White squares indicate positive weights, and black squares indicate negative weights. Size of square indicates magnitude of weight. First 24 rows from bottom up indicate raw board information. Letting x=number of men before the move and y=number of men after the move, the interpretations of columns are as follows:  A: x<=-5; B: x=-4; C: x=-3; D: x<=-2; E: x=-1; F: x=1; G: x>=2; H: x=3; I: x=4; J: x>=5; K: x<1 & y=1; L: x<2 & y>=2; M: x<3 & y=3; N: x<4 & y=4; O: x<y & y>=5; P: x=1 & y=0; Q: x>=2 & y=0; R: x>=2 & y=1; S: x>=3 & y=2; T: x>=4 & y=3; U: x>=5 & y=4; V: x>y & y>=5; W: prob. of a White blot at this location being hit (pre-computed feature). The next row encodes number of men on White and Black bars. The next 3 rows encode roll information. Remaining rows encode various pre-computed features.

Much insight into the basis for the network's judgement of various moves has been gained by actually playing games against it. In fact, one of the most revealing tests of what the network has and has not learned came from a 20-game match played by G.T. against one of the latest generation of networks with 48 hidden units. (A detailed description of the match is given in Ref. 11.) The surprising result of this match was that the network actually won, 11 games to 9. However, a

detailed analysis of the moves played by the network during the match indicates that the network was extremely lucky to have won so many games, and could not reasonably be expected to continue to do so well over a large number of games. Out of the 20 games played, there were 11 in which the network did not make any serious mistakes. The network won 6 out of these 11 games, a result which is quite reasonable. However, in 9 of the 20 games, the network made one or more serious (i.e. potentially fatal) "blunders." The seriousness of these mistakes would be equivalently to dropping a piece in chess. Such a mistake is nearly always fatal in chess against a good opponent; however in backgammon there are still chances due to the element of luck involved. In the 9 games in which the network blundered, it did manage to survive and win 5 of the games due to the element of luck. (We are assuming that the mistakes made by the human, if any, were only minor mistakes.) It is highly unlikely that this sort of result would be repeated. A much more likely result would be that the network would win only one or two of the games in which it made a serious error. This would put the network's expected performance against expert or near-expert humans at about the 35-40% level. (This has also been confirmed in play against other networks.)

We find that the network does act as if it has picked up many of the global concepts and strategies of advanced play. The network has also learned many important tactical elements of play at the advanced level. As for the specific kinds of mistakes made by the network, we find that they are not at all random, senseless mistakes, but instead fall into clear, well-defined conceptual categories, and furthermore, one can understand the reasons why these categories of mistakes are made. We do not have space here to describe these in detail, and refer the reader instead to Ref. 5.

To summarize, qualitative analysis of the network's play indicates that it has learned many important strategies and tactics of advanced backgammon. This gives the network very good overall performance in typical positions. However, the network's worst case performance leaves a great deal to be desired. The network is capable of making both serious, obvious, "blunders," as well more subtle mistakes, in many different types of positions. Worst case performance is important, because the network must make long sequences of moves throughout the course of a game without any serious mistakes in order to have a reasonable chance of winning against a skilled opponent. The prospects for improving the network's worst case performance appear to be mixed. It seems quite likely that many of the current "blunders" can be fixed with a reasonable number of hand-crafted examples added to the training set. However, many of the subtle mistakes are due to a lack of very sophisticated knowledge, such as the notion of timing. It is difficult to imagine that this kind of knowledge could be imparted to the network in only a few examples. Probably what is required is either an intractably large number of examples, or a major overhaul in either the pre-computed features or the training paradigm.

## DISCUSSION

We have seen from both quantitative and qualitative measures that the network has learned a great deal about the general principles of backgammon play, and has not simply memorized the individual positions in the training set. Quantitatively, the measure of game performace provides a clear indication of the network's ability to generalize, because apart from the first couple of moves at the start of each game, the network must operate entirely on generalization. Qualitatively, one can see after playing several games against the network that there are certain characteristic kinds of positions in which it does well, and other kinds of positions in which it systematically makes well-defined types of mistakes. Due to the network's frequent "blunders," its overall level of play is only intermediate level, although it probably is somewhat better than the average intermediate-level player. Against the intermediate-level program Gammontool, our best network wins almost 60% of the games. However, against a human expert the network would only win about 35-40% of the time. Thus while the network does not play at expert level, it is sufficiently good to give an expert a hard time, and with luck in its favor can actually win a match to a small number of games.

Our simple supervised learning approach leaves out some very important sources of

information which are readily available to humans. The network is never told that the underlying topological structure of its input space really corresponds to a one-dimensional spatial structure; all it knows is that the inputs form a 459-dimensional hypercube. It has no idea of the object of the game, nor of the sense of temporal causality, i.e. the notion that its actions have consequences, and how those consequences lead to the achievement of the objective. The teacher signal only says whether a given move is good or bad, without giving any indication as to what the teacher's reasons are for making such a judgement. Finally, the network is only capable of scoring single moves in isolation, without any idea of what other moves are available. These sources of knowledge are essential to the ability of humans to play backgammon well, and it seems likely that some way of incorporating them into the network learning paradigm will be necessary in order to achieve further substantial improvements in performance.

There are a number of ways in which these additional sources of knowledge might be incorporated, and we shall be exploring some of them in future work. For example, knowledge of alternative moves could be introduced by defining a more sophisticated error signal which takes into account not only the network and teacher scores for the current move, but also the network and teacher scores for other moves from the same position. However, the more immediate plans involve a continuation of the existing strategies of hand-crafting examples and coding scheme modifications to eliminate the most serious errors in the network's play. If these errors can be eliminated, and we are confident that this can be achieved, then the network would become substantially better than any commercially available program, and would be a serious challenge for human experts. We would expect 65% performance against Gammontool, and 45% performance against human experts.

Some of the results of our study have implications beyond backgammon to more general classes of difficult problems. One of the limitations we have found is that substantial human effort is required both in the design of the coding scheme and in the design of the training set. It is not sufficient to use a simple coding scheme and random training patterns, and let the automated network learning procedure take care of everything else. We expect this to be generally true when connectionist learning is applied to difficult problem domains.

On the positive side, we foresee a potential for combining connectionist learning techniques with conventional AI techniques for hand-crafting knowledge to make significant progress in the development of intelligent systems. From the practical point of view, network learning can be viewed as an "enhancer" of traditional techniques, which might produce systems with superior performance. For this particular application, the obvious way to combine the two approaches is in the use of pre-computed features in the input encoding. Any set of hand-crafted features used in a conventional evaluation function could be encoded as discrete or continuous activity levels of input units which represent the current board state along with the units representing the raw information. Given a suitable encoding scheme for these features, and a training set of sufficient size and quality (i.e., the scores in the training set should be better than those of the original evaluation function), it seems possible that the resulting network could outperform the original evaluation function, as evidenced by our experiment with the Gammontool features.

Network learning might also hold promise as a means of achieving the long-sought goal of automated feature discovery[2]. Our network certainly appears to have learned a great deal of knowledge from the training set which goes far beyond the amount of knowledge that was explicitly encoded in the input features. Some of this knowledge (primarily the lowest level components) is apparent from the weight diagram when there are no hidden units (Fig. 3). However, much of the network's knowledge remains inaccessible. What is needed now is a means of disentangling the novel features discovered by the network from either the patterns of activity in the hidden units, or from the massive number of connection strengths which characterize the network. This is one our top priorities for future research, although techniques for such "reverse engineering" of parallel networks are only beginning to be developed[12].

## ACKNOWLEDGEMENTS

This work was inspired by a conference on "Evolution, Games and Learning" held at Los Alamos National Laboratory, May 20-24, 1985. We thank Sun Microsystems Inc. for providing the source code for their Gammontool program, Hans Berliner for providing some of the positions used in the data base, Subutai Ahmad for writing the weight display graphics package, Bill Bogstad for assistance in programming the back-propagation simulator, and Bartlett Mel, Peter Frey, and Scott Kirkpatrick for critical reviews of the manuscript. G.T. was supported in part by the National Center for Supercomputing Applications. T.J.S. was supported by a NSF Presidential Young Investigator Award, and by grants from the Seaver Institute and the Lounsbury Foundation.

## REFERENCES

1. D. E. Rumelart and J. L. McClelland, eds., *Parallel Distributed Processing: Explorations in the Microstructure of Cognition*, Vols. 1 and 2 (Cambridge: MIT Press, 1986).

2. A. L. Samuel, "Some studies in machine learning using the game of checkers." *IBM J. Res. Dev.* 3, 210--229 (1959).

3. J. H. Holland, "Escaping brittleness: the possibilities of general-purpose learning algorithms applied to parallel rule-based systems." In: R. S. Michalski et al. (eds.), *Machine learning: an artificial intelligence approach, Vol. II* (Los Altos CA: Morgan-Kaufman, 1986).

4. R. S. Sutton, "Learning to predict by the methods of temporal differences," GTE Labs Tech. Report TR87-509.1 (1987).

5. G. Tesauro and T. J. Sejnowski, "A parallel network that learns to play backgammon." Univ. of Illinois at Urbana-Champaign, Center for Complex Systems Research Technical Report (1987).

6. M. Minsky and S. Papert, *Perceptrons* (Cambridge: MIT Press, 1969).

7. D. E. Rumelhart, G. E. Hinton, and R. J. Williams, "Learning representations by back-propagating errors." *Nature* 323, 533--536 (1986).

8. Y. Le Cun, "A learning procedure for asymmetric network." *Proceedings of Cognitiva (Paris)* 85, 599--604 (1985).

9. D. B. Parker, "Learning-logic." MIT Center for Computational Research in Economics and Management Science Tech. Report TR-47 (1985).

10. H. Berliner, "Backgammon computer program beats world champion." *Artificial Intelligence* 14, 205--220 (1980).

11. G. Tesauro, "Neural network defeats creator in backgammon match." Univ. of Illinois at Urbana-Champaign, Center for Complex Systems Research Technical Report (1987).

12. C. R. Rosenberg, "Revealing the structure of NETtalk's internal representations." Proceedings of the Ninth Annual Conference of the Cognitive Science Society (Hillsdale, NJ: Lawrence Erlbaum Associates, 1987).